# Local probability propagation for factor analysis

**Brendan J. Frey**

Computer Science, University of Waterloo, Waterloo, Ontario, Canada

## Abstract

Ever since Pearl's probability propagation algorithm in graphs with cycles was shown to produce excellent results for error-correcting decoding a few years ago, we have been curious about whether local probability propagation could be used successfully for machine learning. One of the simplest adaptive models is the factor analyzer, which is a two-layer network that models bottom layer sensory inputs as a linear combination of top layer factors plus independent Gaussian sensor noise. We show that local probability propagation in the factor analyzer network usually takes just a few iterations to perform accurate inference, even in networks with 320 sensors and 80 factors. We derive an expression for the algorithm's fixed point and show that this fixed point matches the exact solution in a variety of networks, even when the fixed point is unstable. We also show that this method can be used successfully to perform inference for approximate EM and we give results on an online face recognition task.

## 1 Factor analysis

A simple way to encode input patterns is to suppose that each input can be well-approximated by a linear combination of component vectors, where the amplitudes of the vectors are modulated to match the input. For a given training set, the most appropriate set of component vectors will depend on how we expect the modulation levels to behave and how we measure the distance between the input and its approximation. These effects can be captured by a generative probability model that specifies a distribution $p(\mathbf{z})$ over modulation levels $\mathbf{z} = (z_1, \ldots, z_K)^{\mathrm{T}}$ and a distribution $p(\mathbf{x}|\mathbf{z})$ over sensors $\mathbf{x} = (x_1, \ldots, x_N)^{\mathrm{T}}$ given the modulation levels. Principal component analysis, independent component analysis and factor analysis can be viewed as maximum likelihood learning in a model of this type, where we assume that over the training set, the appropriate modulation levels are independent and the overall distortion is given by the sum of the individual sensor distortions.

In factor analysis, the modulation levels are called *factors* and the distributions have the following form:

$$p(z_k) = \mathcal{N}(z_k; 0, 1), \quad p(\mathbf{z}) = \prod_{k=1}^{K} p(z_k) = \mathcal{N}(\mathbf{z}; \mathbf{0}, \mathbf{I}),$$

$$p(x_n|\mathbf{z}) = \mathcal{N}(x_n; \sum_{k=1}^{K} \lambda_{nk} z_k, \psi_n), \quad p(\mathbf{x}|\mathbf{z}) = \prod_{n=1}^{N} p(x_n|\mathbf{z}) = \mathcal{N}(\mathbf{x}; \mathbf{\Lambda z}, \mathbf{\Psi}). \quad (1)$$

The parameters of this model are the factor loading matrix $\mathbf{\Lambda}$, with elements $\lambda_{nk}$, and the diagonal sensor noise covariance matrix $\mathbf{\Psi}$, with diagonal elements $\psi_n$. A belief network for the factor analyzer is shown in Fig. 1a. The likelihood is

$$p(\mathbf{x}) = \int_{\mathbf{z}} \mathcal{N}(\mathbf{z}; \mathbf{0}, \mathbf{I}) \mathcal{N}(\mathbf{x}; \mathbf{\Lambda z}, \mathbf{\Psi}) d\mathbf{z} = \mathcal{N}(\mathbf{x}; \mathbf{0}, \mathbf{\Lambda \Lambda}^{\mathrm{T}} + \mathbf{\Psi}), \quad (2)$$

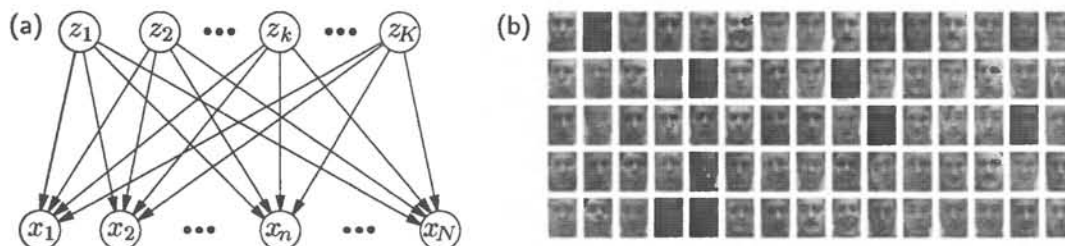

Figure 1: (a) A belief network for factor analysis. (b) High-dimensional data ($N = 560$).

and online factor analysis consists of adapting $\mathbf{\Lambda}$ and $\mathbf{\Psi}$ to increase the likelihood of the current input, such as a vector of pixels from an image in Fig. 1b.

Probabilistic inference – computing or estimating $p(\mathbf{z}|\mathbf{x})$ – is needed to do dimensionality reduction and to fill in the unobserved factors for online EM-type learning. In this paper, we focus on methods that infer *independent* factors. $p(\mathbf{z}|\mathbf{x})$ is Gaussian and it turns out that the posterior means and variances of the factors are

$$\mathrm{E}[\mathbf{z}|\mathbf{x}] = (\mathbf{\Lambda}^{\mathrm{T}}\mathbf{\Psi}^{-1}\mathbf{\Lambda} + \mathbf{I})^{-1}\mathbf{\Lambda}^{\mathrm{T}}\mathbf{\Psi}^{-1}\mathbf{x},$$
$$\mathrm{diag}(\mathrm{COV}(\mathbf{z}|\mathbf{x})) = \mathrm{diag}((\mathbf{\Lambda}^{\mathrm{T}}\mathbf{\Psi}^{-1}\mathbf{\Lambda} + \mathbf{I})^{-1}). \tag{3}$$

Given $\mathbf{\Lambda}$ and $\mathbf{\Psi}$, computing these values exactly takes $\mathcal{O}(K^2 N)$ computations, mainly because of the time needed to compute $\mathbf{\Lambda}^{\mathrm{T}}\mathbf{\Psi}^{-1}\mathbf{\Lambda}$. Since there are only $KN$ connections in the network, exact inference takes at least $\mathcal{O}(K)$ bottom-up/top down iterations.

Of course, if the *same* network is going to be applied more than $K$ times for inference (*e.g.*, for batch EM), then the matrices in (3) can be computed once and reused. However, this is not directly applicable in online learning and in biological models. One way to circumvent computing the matrices is to keep a separate recognition network, which approximates $\mathrm{E}[\mathbf{z}|\mathbf{x}]$ with $\mathbf{R}\mathbf{x}$ (Dayan *et al.*, 1995). The optimal recognition network, $\mathbf{R} = (\mathbf{\Lambda}^{\mathrm{T}}\mathbf{\Psi}^{-1}\mathbf{\Lambda}+\mathbf{I})^{-1}\mathbf{\Lambda}^{\mathrm{T}}\mathbf{\Psi}^{-1}$, can be approximated by jointly estimating the generative network and the recognition network using online wake-sleep learning (Hinton *et al.*, 1995).

## 2   Probability propagation in the factor analyzer network

Recent results on error-correcting coding show that in some cases Pearl's probability propagation algorithm, which does exact probabilistic inference in graphs that are trees, gives excellent performance *even if the network contains so many cycles that its minimal cut set is exponential* (Frey and MacKay, 1998; Frey, 1998; MacKay, 1999). In fact, the probability propagation algorithm for decoding low-density parity-check codes (MacKay, 1999) and turbocodes (Berrou and Glavieux, 1996) is widely considered to be a major breakthrough in the information theory community.

When the network contains cycles, the local computations give rise to an iterative algorithm, which hopefully converges to a good answer. Little is known about the convergence properties of the algorithm. Networks containing a single cycle have been successfully analyzed by Weiss (1999) and Smyth *et al.* (1997), but results for networks containing many cycles are much less revealing.

The probability messages produced by probability propagation in the factor analyzer network of Fig. 1a are Gaussians. Each iteration of propagation consists of passing a mean and a variance along each edge in a bottom-up pass, followed by passing a mean and a variance along each edge in a top-down pass. At any instant, the

bottom-up means and variances can be combined to form estimates of the means and variances of the modulation levels given the input.

Initially, the variance and mean sent from the $k$th top layer unit to the $n$th sensor is set to $\nu_{kn}^{(0)} = 1$ and $\eta_{kn}^{(0)} = 0$. The bottom-up pass begins by computing a noise level and an error signal at each sensor using the top-down variances and means from the previous iteration:

$$s_n^{(i)} = \psi_n + \sum_{k=1}^{K} \lambda_{nk}^2 \nu_{kn}^{(i-1)}, \qquad e_n^{(i)} = x_n - \sum_{k=1}^{K} \lambda_{nk} \eta_{kn}^{(i-1)}. \qquad (4)$$

These are used to compute bottom-up variances and means as follows:

$$\phi_{nk}^{(i)} = s_n^{(i)}/\lambda_{nk}^2 - \nu_{kn}^{(i-1)}, \qquad \mu_{nk}^{(i)} = e_n^{(i)}/\lambda_{nk} + \eta_{kn}^{(i-1)}. \qquad (5)$$

The bottom-up variances and means are then combined to form the current estimates of the modulation variances and means:

$$v_k^{(i)} = 1/(1 + \sum_{n=1}^{N} 1/\phi_{nk}^{(i)}), \qquad \hat{z}_k^{(i)} = v_k^{(i)} \sum_{n=1}^{N} \mu_{nk}^{(i)}/\phi_{nk}^{(i)}. \qquad (6)$$

The top-down pass proceeds by computing top-down variances and means as follows:

$$\nu_{kn}^{(i)} = 1/(1/v_k^{(i)} - 1/\phi_{nk}^{(i)}), \qquad \eta_{kn}^{(i)} = \nu_{kn}^{(i)}(\hat{z}_k^{(i)}/v_k^{(i)} - \mu_{nk}^{(i)}/\phi_{nk}^{(i)}). \qquad (7)$$

Notice that the variance updates are independent of the mean updates, whereas the mean updates depend on the variance updates.

**2.1  Performance of local probability propagation.**   We created a total of 200,000 factor analysis networks with 20 different sizes ranging from $K = 5$, $N = 10$ to $K = 80$, $N = 320$ and for each size of network we measured the inference error as a function of the number of iterations of propagation. Each of the 10,000 networks of a given size was produced by drawing the $\lambda_{nk}$s from standard normal distributions and then drawing each sensor variance $\psi_n$ from an exponential distribution with mean $\sum_{k=1}^{K} \lambda_{nk}^2$. (A similar procedure was used by Neal and Dayan (1997).)

For each random network, a pattern was simulated from the network and probability propagation was applied using the simulated pattern as input. We measured the error between the estimate $\hat{\mathbf{z}}^{(i)}$ and the correct value $E[\mathbf{z}|\mathbf{x}]$ by computing the difference between their coding costs under the exact posterior distribution and then normalizing by $K$ to get an average number of nats per top layer unit.

Fig. 2a shows the inference error on a logarithmic scale versus the number of iterations (maximum of 20) in the 20 different network sizes. In all cases, the median error is reduced below .01 nats within 6 iterations. The rate of convergence of the error improves for larger $N$, as indicated by a general trend for the error curves to drop when $N$ is increased. In contrast, the rate of convergence of the error appears to worsen for larger $K$, as shown by a general slight trend for the error curves to rise when $K$ is increased.

For $K \geq N/8$, 0.1% of the networks actually diverge. To better understand the divergent cases, we studied the means and variances for all of the divergent networks. In all cases, the variances converge within a few iterations whereas the means oscillate and diverge. For $K = 5$, $N = 10$, 54 of the 10,000 networks diverged and 5 of these are shown in Fig. 2b. This observation suggests that in general the dynamics are determined by the dynamics of the mean updates.

**2.2  Fixed points and a condition for global convergence.**   When the variance updates converge, the dynamics of probability propagation in factor analysis networks become linear. This allows us to derive the fixed point of propagation in closed form and write an eigenvalue condition for global convergence.

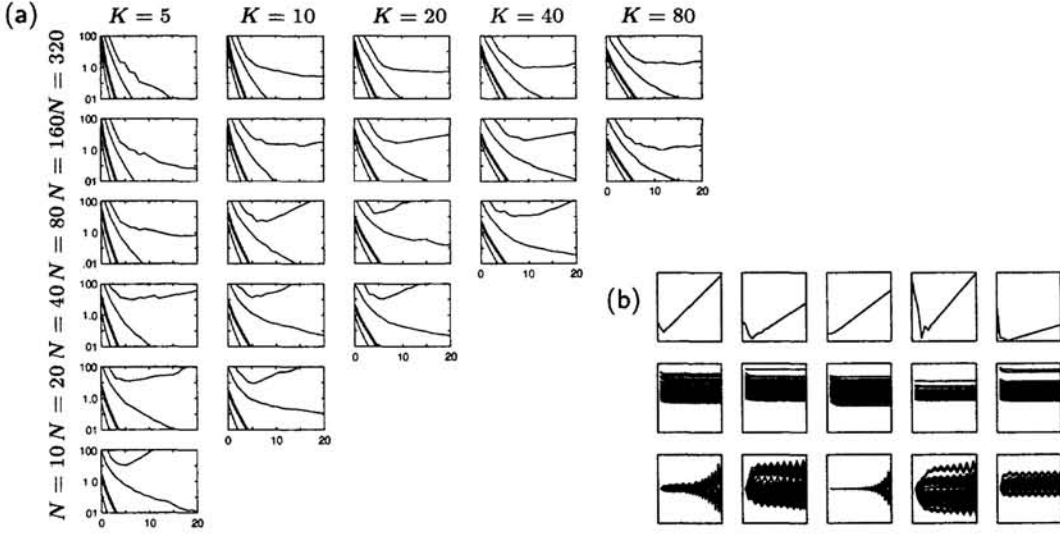

Figure 2: (a) Performance of probability propagation. Median inference error (bold curve) on a logarithmic scale as a function of the number of iterations for different sizes of network parameterized by $K$ and $N$. The two curves adjacent to the bold curve show the range within which 98% of the errors lie. 99.9% of the errors were below the fourth, topmost curve. (b) The error, bottom-up variances and top-down means as a function of the number of iterations (maximum of 20) for 5 divergent networks of size $K = 5$, $N = 10$.

To analyze the system of mean updates, we define the following length $KN$ vectors of means and the input: $\tilde{\boldsymbol{\eta}}^{(i)} = (\eta_{11}^{(i)}, \eta_{21}^{(i)}, \ldots, \eta_{K1}^{(i)}, \eta_{12}^{(i)}, \ldots, \eta_{KN}^{(i)})^{\mathrm{T}}$, $\tilde{\boldsymbol{\mu}}^{(i)} = (\mu_{11}^{(i)}, \mu_{12}^{(i)}, \ldots, \mu_{1K}^{(i)}, \mu_{21}^{(i)}, \ldots, \mu_{NK}^{(i)})^{\mathrm{T}}$, $\tilde{\mathbf{x}} = (x_1, x_1, \ldots, x_1, x_2, \ldots, x_2, x_N, \ldots, x_N)^{\mathrm{T}}$, where each $x_n$ is repeated $K$ times in the last vector. The network parameters are represented using $KN \times KN$ diagonal matrices, $\tilde{\boldsymbol{\Lambda}}$ and $\tilde{\boldsymbol{\Psi}}$. The diagonal of $\tilde{\boldsymbol{\Lambda}}$ is $\lambda_{11}, \ldots, \lambda_{1K}, \lambda_{21}, \ldots, \lambda_{NK}$, and the diagonal of $\tilde{\boldsymbol{\Psi}}$ is $\psi_1 \mathbf{I}, \psi_2 \mathbf{I}, \ldots, \psi_N \mathbf{I}$, where $\mathbf{I}$ is the $K \times K$ identity matrix. The converged bottom-up variances are represented using a diagonal matrix $\tilde{\boldsymbol{\Phi}}$ with diagonal $\phi_{11}, \ldots, \phi_{1K}, \phi_{21}, \ldots, \phi_{NK}$.

The summation operations in the propagation formulas are represented by a $KN \times KN$ matrix $\tilde{\boldsymbol{\Sigma}}_z$ that sums over means sent down from the top layer and a $KN \times KN$ matrix $\tilde{\boldsymbol{\Sigma}}_x$ that sums over means sent up from the sensory input:

$$\tilde{\boldsymbol{\Sigma}}_z = \begin{pmatrix} \mathbf{1} & & & \\ & \mathbf{1} & & \\ & & \ldots & \\ & & & \mathbf{1} \end{pmatrix}, \quad \tilde{\boldsymbol{\Sigma}}_x = \begin{pmatrix} \mathbf{I} & \mathbf{I} & \ldots & \mathbf{I} \\ \mathbf{I} & \mathbf{I} & \ldots & \mathbf{I} \\ \ldots & \ldots & \ldots & \ldots \\ \mathbf{I} & \mathbf{I} & \ldots & \mathbf{I} \end{pmatrix}. \tag{8}$$

These are $N \times N$ matrices of $K \times K$ blocks, where $\mathbf{1}$ is the $K \times K$ block of ones and $\mathbf{I}$ is the $K \times K$ identity matrix.

Using the above representations, the bottom-up pass is given by

$$\tilde{\boldsymbol{\mu}}^{(i)} = \tilde{\boldsymbol{\Lambda}}^{-1}\tilde{\mathbf{x}} - \tilde{\boldsymbol{\Lambda}}^{-1}(\tilde{\boldsymbol{\Sigma}}_z - \mathbf{I})\tilde{\boldsymbol{\Lambda}}\tilde{\boldsymbol{\eta}}^{(i-1)}, \tag{9}$$

and the top-down pass is given by

$$\tilde{\boldsymbol{\eta}}^{(i)} = \left(\mathbf{I} + \mathrm{diag}(\tilde{\boldsymbol{\Sigma}}_x\tilde{\boldsymbol{\Phi}}^{-1}\tilde{\boldsymbol{\Sigma}}_x) - \tilde{\boldsymbol{\Phi}}^{-1}\right)^{-1}(\tilde{\boldsymbol{\Sigma}}_x - \mathbf{I})\tilde{\boldsymbol{\Phi}}^{-1}\tilde{\boldsymbol{\mu}}^{(i)}. \tag{10}$$

Substituting (10) into (9), we get the linear update for $\tilde{\boldsymbol{\mu}}$:

$$\tilde{\boldsymbol{\mu}}^{(i)} = \tilde{\boldsymbol{\Lambda}}^{-1}\tilde{\mathbf{x}} - \tilde{\boldsymbol{\Lambda}}^{-1}(\tilde{\boldsymbol{\Sigma}}_z - \mathbf{I})\tilde{\boldsymbol{\Lambda}}\left(\mathbf{I} + \mathrm{diag}(\tilde{\boldsymbol{\Sigma}}_x\tilde{\boldsymbol{\Phi}}^{-1}\tilde{\boldsymbol{\Sigma}}_x) - \tilde{\boldsymbol{\Phi}}^{-1}\right)^{-1}(\tilde{\boldsymbol{\Sigma}}_x - \mathbf{I})\tilde{\boldsymbol{\Phi}}^{-1}\tilde{\boldsymbol{\mu}}^{(i-1)}. \tag{11}$$

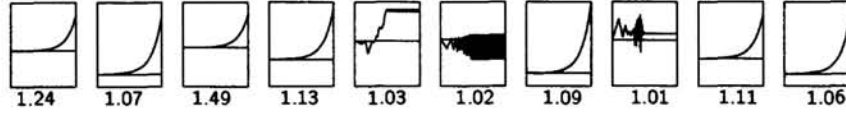

Figure 3: The error (log scale) versus number of iterations (log scale, max. of 1000) in 10 of the divergent networks with $K = 5$, $N = 10$. The means were *initialized* to the fixed point solutions and machine round-off errors cause divergence from the fixed points, whose errors are shown by horizontal lines.

The fixed point of this dynamic system, when it exists, is

$$\tilde{\mu}^* = \tilde{\Phi} \left( \tilde{\Lambda}\tilde{\Phi} + (\tilde{\Sigma}_z - \mathbf{I})\tilde{\Lambda} (\mathbf{I} + \text{diag}(\tilde{\Sigma}_x \tilde{\Phi}^{-1} \tilde{\Sigma}_x) - \tilde{\Phi}^{-1})^{-1} (\tilde{\Sigma}_x - \mathbf{I}) \right)^{-1} \tilde{\mathbf{x}}. \quad (12)$$

A fixed point exists if the determinant of the expression in large braces in (12) is nonzero. We have found a simplified expression for this determinant in terms of the determinants of smaller, $K \times K$ matrices.

Reinterpreting the dynamics in (11) as dynamics for $\tilde{\Lambda}\tilde{\mu}^{(i)}$, the stability of a fixed point is determined by the largest eigenvalue of the update matrix, $(\tilde{\Sigma}_z - \mathbf{I})\tilde{\Lambda}(\mathbf{I} + \text{diag}(\tilde{\Sigma}_x \tilde{\Phi}^{-1} \tilde{\Sigma}_x) - \tilde{\Phi}^{-1})^{-1}(\tilde{\Sigma}_x - \mathbf{I})\tilde{\Phi}^{-1}\tilde{\Lambda}^{-1}$. If the modulus of the largest eigenvalue is less than 1, the fixed point is stable. Since the system is linear, if a stable fixed point exists, the system will be globally convergent to this point.

Of the 200,000 networks we explored, about 99.9% of the networks converged. For 10 of the divergent networks with $K = 5$, $N = 10$, we used 1000 iterations of probability propagation to compute the steady state variances. Then, we computed the modulus of the largest eigenvalue of the system and we computed the fixed point. After initializing the bottom-up means to the fixed point values, we performed 1000 iterations to see if numerical errors due to machine precision would cause divergence from the fixed point. Fig. 3 shows the error versus number of iterations (on logarithmic scales) for each network, the error of the fixed point, and the modulus of the largest eigenvalue. In some cases, the network diverges from the fixed point and reaches a dynamic equilibrium that has a lower average error than the fixed point.

## 3   Online factor analysis

To perform maximum likelihood factor analysis in an online fashion, each parameter should be modified to slightly increase the log-probability of the current sensory input, $\log p(\mathbf{x})$. However, since the factors are hidden, they must be probabilistically "filled in" using inference before an incremental learning step is performed.

If the estimated mean and variance of the $k$th factor are $\hat{z}_k$ and $v_k$, then it turns out (*e.g.*, Neal and Dayan, 1997) the parameters can be updated as follows:

$$\lambda_{nk} \leftarrow \lambda_{nk} + \eta[\hat{z}_k(x_n - \sum_{j=1}^{K}\lambda_{nj}\hat{z}_j) - v_k\lambda_{nk}]/\psi_n,$$

$$\psi_n \leftarrow (1 - \eta)\psi_n + \eta[(x_n - \sum_{j=1}^{K}\lambda_{nj}\hat{z}_j)^2 + \sum_{j=1}^{K}v_k\lambda_{nj}^2], \quad (13)$$

where $\eta$ is a learning rate.

Online learning consists of performing some number of iterations of probability propagation for the current input (*e.g.*, 4 iterations) and then modifying the parameters before processing the next input.

**3.1   Results on simulated data.**   We produced 95 training sets of 200 cases each, with input sizes ranging from 20 sensors to 320 sensors. For each of 19 sizes of factor analyzer, we randomly selected 5 sets of parameters as described above and generated a training set. The factor analyzer sizes were $K \in \{5, 10, 20, 40, 80\}$,

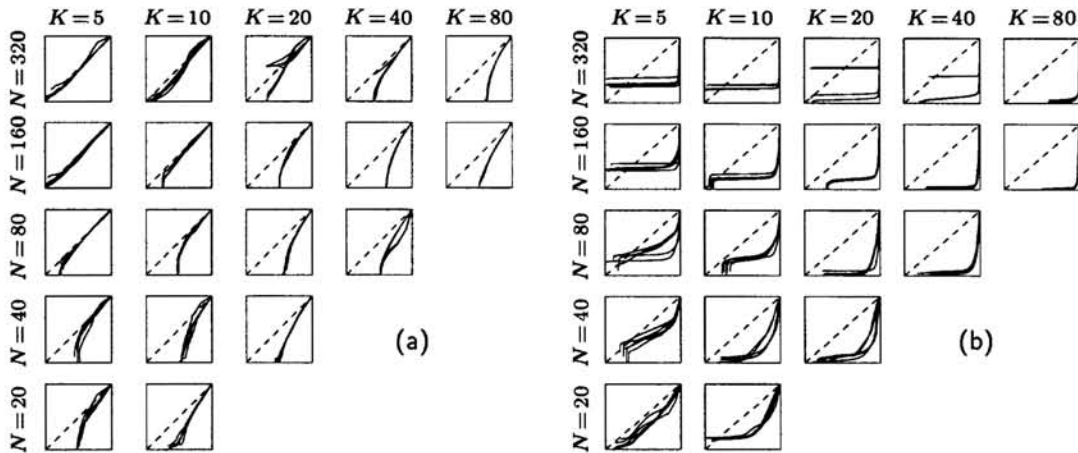

Figure 4: (a) Achievable errors after the same number of epochs of learning using 4 iterations versus 1 iteration. The horizontal axis gives the log-probability error (log scale) for learning with 1 iteration and the vertical axis gives the error after the same number of epochs for learning with 4 iterations. (b) The achievable errors for learning using 4 iterations of propagation versus wake-sleep learning using 4 iterations.

$N \in \{20, 40, 80, 160, 320\}$, $N > K$. For each factor analyzer and simulated data set, we estimated the optimal log-probability of the data using 100 iterations of EM.

For learning, the size of the model to be trained was set equal to the size of the model that was used to generate the data. To avoid the issue of how to schedule learning rates, we searched for *achievable* learning curves, regardless of whether or not a simple schedule for the learning rate exists. So, for a given method and randomly initialized parameters, we performed one separate epoch of learning using each of the learning rates, $1, 0.5, \dots, 0.5^{20}$ and picked the learning rate that most improved the log-probability. Each successive learning rate was determined by comparing the performance using the old learning rate and one 0.75 times smaller.

We are mainly interested in comparing the achievable curves for different methods and how the differences scale with $K$ and $N$. For two methods with the same $K$ and $N$ trained on the same data, we plot the log-probability error (optimal log-probability minus log-probability under the learned model) of one method against the log-probability error of the other method.

Fig. 4a shows the achievable errors using 4 iterations versus using 1 iteration. Usually, using 4 iterations produces networks with lower errors than those learned using 1 iteration. The difference is most significant for networks with large $K$, where in Sec. 2.1 we found that the convergence of the inference error was slower.

Fig. 4b shows the achievable errors for learning using 4 iterations of probability propagation versus wake-sleep learning using 4 iterations. Generally, probability propagation achieves much smaller errors than wake-sleep learning, although for small $K$ wake-sleep performs better very close to the optimum log-probability. The most significant difference between the methods occurs for large $K$, where aside from local optima probability propagation achieves nearly optimal log-probabilities while the log-probabilities for wake-sleep learning are still close to their values at the start of learning.

## 4  Online face recognition

Fig. 1b shows examples from a set of 30,000 $20 \times 28$ greyscale face images of 18 different people. In contrast to other data sets used to test face recognition methods, these faces include wide variation in expression and pose. To make classification more difficult, we normalized the images for each person so that each pixel has

the same mean and variance. We used probability propagation and a recognition network in a factor analyzer to reduce the dimensionality of the data online from 560 dimensions to 40 dimensions. For probability propagation, we rather arbitrarily chose a learning rate of 0.0001, but for wake-sleep learning we tried learning rates ranging from 0.1 down to 0.0001. A multilayer perceptron with one hidden layer of 160 tanh units and one output layer of 18 softmax units was simultaneously being trained using gradient descent to predict face identity from the mean factors. The learning rate for the multilayer perceptron was set to 0.05 and this value was used for both methods.

For each image, a prediction was made before the parameters were modified. Fig. 5 shows online error curves obtained by filtering the losses. The curve for probability propagation is generally below the curves for wake-sleep learning.

The figure also shows the error curves for two forms of online nearest neighbors, where only the most recent $W$ cases are used to make a prediction. The form of nearest neighbors that performs the worst has $W$ set so that the storage requirements are the same as for the factor analysis / multilayer perceptron method. The better form of nearest neighbors has $W$ set so that the number of computations is the same as for the factor analysis / multilayer perceptron method.

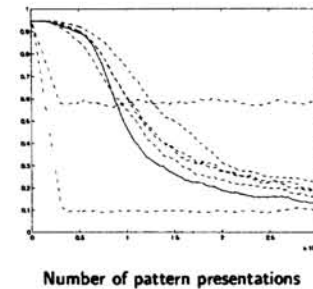

Number of pattern presentations

Figure 5: Online error curves for probability propagation (solid), wake-sleep learning (dashed), nearest neighbors (dot-dashed) and guessing (dotted).

## 5 Summary

It turns out that iterative probability propagation can be fruitful when used for learning in a graphical model with cycles, even when the model is *densely* connected. Although we are more interested in extending this work to more complex models where exact inference takes exponential time, studying iterative probability propagation in the factor analyzer allowed us to compare our results with exact inference and allowed us to derive the fixed point of the algorithm. We are currently applying iterative propagation in multiple cause networks for vision problems.

### References

C. Berrou and A. Glavieux 1996. Near optimum error correcting coding and decoding: Turbo-codes. *IEEE Trans. on Communications*, **44**, 1261–1271.

P. Dayan, G. E. Hinton, R. M. Neal and R. S. Zemel 1995. The Helmholtz machine. *Neural Computation* **7**, 889–904.

B. J. Frey and D. J. C. MacKay 1998. A revolution: Belief propagation in graphs with cycles. In M. Jordan, M. Kearns and S. Solla (eds), *Advances in Neural Information Processing Systems 10*, Denver, 1997.

B. J. Frey 1998. *Graphical Models for Machine Learning and Digital Communication.* MIT Press, Cambridge MA. See http://www.cs.utoronto.ca/~frey.

G. E. Hinton, P. Dayan, B. J. Frey and R. M. Neal 1995. The wake-sleep algorithm for unsupervised neural networks. *Science* **268**, 1158–1161.

D. J. C. MacKay 1999. *Information Theory, Inference and Learning Algorithms.* Book in preparation, currently available at http://wol.ra.phy.cam.ac.uk/mackay.

R. M. Neal and P. Dayan 1997. Factor analysis using delta-rule wake-sleep learning. *Neural Computation* **9**, 1781–1804.

P. Smyth, R. J. McEliece, M. Xu, S. Aji and G. Horn 1997. Probability propagation in graphs with cycles. Presented at the workshop on *Inference and Learning in Graphical Models*, Vail, Colorado.

Y. Weiss 1998. Correctness of local probability propagation in graphical models. To appear in *Neural Computation*.
